# Regularization with Dot-Product Kernels

**Alex J. Smola, Zoltán L. Óvári, and Robert C. Williamson**
Department of Engineering
Australian National University
Canberra, ACT, 0200

## Abstract

In this paper we give necessary and sufficient conditions under which kernels of dot product type $k(x,y) = k(x \cdot y)$ satisfy Mercer's condition and thus may be used in Support Vector Machines (SVM), Regularization Networks (RN) or Gaussian Processes (GP). In particular, we show that if the kernel is analytic (i.e. can be expanded in a Taylor series), all expansion coefficients have to be nonnegative. We give an explicit functional form for the feature map by calculating its eigenfunctions and eigenvalues.

## 1 Introduction

Kernel functions are widely used in learning algorithms such as Support Vector Machines, Gaussian Processes, or Regularization Networks. A possible interpretation of their effects is that they represent dot products in some *feature space* $\mathcal{F}$, i.e.

$$k(x,y) = \phi(x) \cdot \phi(y) \tag{1}$$

where $\phi$ is a map from input (data) space $\mathcal{X}$ into $\mathcal{F}$. Another interpretation is to connect $\phi$ with the regularization properties of the corresponding learning algorithm [8]. Most popular kernels can be described by three main categories: translation invariant kernels [9]

$$k(x,y) = k(x - y), \tag{2}$$

kernels originating from generative models (e.g. those of Jaakkola and Haussler, or Watkins), and thirdly, dot-product kernels

$$k(x,y) = k(x \cdot y). \tag{3}$$

Since $k$ influences the properties of the estimates generated by any of the algorithms above, it is natural to ask which regularization properties are associated with $k$.

In [8, 10, 9] the general connections between kernels and regularization properties are pointed out, containing details on the connection between the Fourier spectrum of translation invariant kernels and the smoothness properties of the estimates. In a nutshell, the necessary and sufficient condition for $k(x - y)$ to be a Mercer kernel (i.e. be admissible for any of the aforementioned kernel methods) is that its Fourier transform be nonnegative. This also allowed for an easy to check criterion for new kernel functions. Moreover, [5] gave a similar analysis for kernels derived from generative models.

Dot product kernels $k(x \cdot y)$, on the other hand, have been eluding further theoretical analysis and only a necessary condition [1] was found, based on geometrical considerations. Unfortunately, it does not provide much insight into smoothness properties of the corresponding estimate.

Our aim in the present paper is to shed some light on the properties of dot product kernels, give an explicit equation how its eigenvalues can be determined, and, finally, show that for analytic kernels that can be expanded in terms of monomials $\xi^n$ or associated Legendre polynomials $P_n^d(\xi)$ [4], i.e.

$$k(x, y) = k(x \cdot y) \text{ with } k(\xi) = \sum_{n=0}^{\infty} a_n \xi^n \text{ or } k(\xi) = \sum_{n=0}^{\infty} b_n P_n^d(\xi) \qquad (4)$$

a necessary and sufficient condition is $a_n \geq 0$ for all $n \in \mathbb{N}$ if no assumption about the dimensionality of the input space is made (for finite dimensional spaces of dimension $d$, the condition is that $b_n \geq 0$). In other words, the polynomial series expansion in dot product kernels plays the role of the Fourier transform in translation invariant kernels.

## 2 Regularization, Kernels, and Integral Operators

Let us briefly review some results from regularization theory, needed for the further understanding of the paper. Many algorithms (SVM, GP, RN, etc.) can be understood as minimizing a regularized risk functional

$$R_{\text{reg}}[f] := R_{\text{emp}}[f] + \lambda \Omega[f] \qquad (5)$$

where $R_{\text{emp}}$ is the *training error* of the function $f$ on the given data, $\lambda > 0$ and $\Omega[f]$ is the so-called *regularization term*. The first term depends on the specific problem at hand (classification, regression, large margin algorithms, etc.), $\lambda$ is generally adjusted by some model selection criterion, and $\Omega[f]$ is a nonnegative functional of $f$ which models our *belief* which functions should be considered to be *simple* (a *prior* in the Bayesian sense or a *structure* in a Structural Risk Minimization sense).

### 2.1 Regularization Operators

One possible interpretation of $k$ is [8] that it leads to regularized risk functionals where

$$\Omega[f] = \frac{1}{2} \|Pf\|^2 \text{ or equivalently } \langle Pk(x, \cdot), Pk(y, \cdot) \rangle = k(x, y). \qquad (6)$$

Here $P$ is a regularization operator mapping functions $f$ on $\mathcal{X}$ into a dot product space (we choose $L_2(\mathcal{X})$). The following theorem allows us to construct explicit operators $P$ and it provides a criterion whether a symmetric function $k(x, y)$ is suitable.

**Theorem 1 (Mercer [3])** *Suppose $k \in L_\infty(\mathcal{X}^2)$ such that the integral operator $T_k : L_2(\mathcal{X}) \to L_2(\mathcal{X})$,*

$$T_k f(\cdot) := \int_{\mathcal{X}} k(\cdot, x) f(x) d\mu(x) \qquad (7)$$

*is positive. Let $\Phi_j \in L_2(\mathcal{X})$ be the eigenfunction of $T_k$ with eigenvalue $\lambda_j \neq 0$ and normalized such that $\|\Phi_j\|_{L_2} = 1$ and let $\overline{\Phi_j}$ denote its complex conjugate. Then*

*1. $(\lambda_j(T))_j \in \ell_1$.*

*2. $\Phi_j \in L_\infty(\mathcal{X})$ and $\sup_j \|\Phi_j\|_{L_\infty} < \infty$.*

*3.* $k(x, x') = \sum\limits_{j \in \mathbb{N}} \lambda_j \overline{\Phi_j(x)} \Phi_j(x')$ *holds for almost all* $(x, x')$, *where the series*

*converges absolutely and uniformly for almost all* $(x, x')$.

This means that by finding the eigensystem $(\lambda_i, \Phi_i)$ of $T_k$ we can also determine the regularization operator $P$ via [8]

$$Pf = \sum_{i=1}^{\infty} \frac{a_i}{\sqrt{\lambda_i}} \Phi_i \text{ for any } f = \sum_{i=1}^{\infty} a_i \Phi_i. \tag{8}$$

The eigensystem $(\lambda_i, \Phi_i)$ tells us which functions are considered "simple" in terms of the operator $P$. Consequently, in order to determine the regularization properties of dot product kernels we have to find their eigenfunctions and eigenvalues.

## 2.2 Specific Assumptions

Before we diagonalize $T_k$ for a given kernel we have yet to specify the assumptions we make about the measure $\mu$ and the domain of integration $\mathcal{X}$. Since a suitable choice can drastically simplify the problem we try to keep as much of the symmetries imposed by $k(x \cdot y)$ as possible. The predominant symmetry in dot product kernels is rotation invariance. Therefore we set choose the *unit ball* in $\mathbb{R}^d$

$$\mathcal{X} := U_d := \{x | x \in \mathbb{R}^d \text{ and } \|x\|_2 \leq 1\}. \tag{9}$$

This is a benign assumption since the radius can always be adjusted by rescaling $k(x \cdot y) \to k((\theta x) \cdot (\theta y))$. Similar considerations apply to translation. In some cases the *unit sphere* in $\mathbb{R}^d$ is more amenable to our analysis. There we choose

$$\mathcal{X} := S_{d-1} := \{x | x \in \mathbb{R}^d \text{ and } \|x\|_2 = 1\}. \tag{10}$$

The latter is a good approximation of the situation where dot product kernels perform best — if the training data has approximately equal Euclidean norm (e.g. in images or handwritten digits). For the sake of simplicity we will limit ourselves to (10) in most of the cases.

Secondly we choose $\mu$ to be the uniform measure on $\mathcal{X}$. This means that we have to solve the following integral equation: Find functions $\Phi_i : L_2(\mathcal{X}) \to \mathbb{R}$ together with coefficients $\lambda_i$ such that $T_k \Phi_i(x) := \int_{\mathcal{X}} k(x \cdot y) \Phi_i(y) dy = \lambda_i \Phi_i(x)$.

## 3 Orthogonal Polynomials and Spherical Harmonics

Before we can give eigenfunctions or state necessary and sufficient conditions we need some basic relations about Legendre Polynomials and spherical harmonics.

Denote by $P_n(\xi)$ the Legendre Polynomials and by $P_n^d(\xi)$ the *associated* Legendre Polynomials (see e.g. [4] for details). They have the following properties

- The polynomials $P_n(\xi)$ and $P_n^d(\xi)$ are of degree $n$, and moreover $P_n := P_n^3$
- The (associated) Legendre Polynomials form an orthogonal basis with

$$\int_{-1}^{1} P_n^d(\xi) P_m^d(\xi) (1 - \xi^2)^{\frac{d-3}{2}} d\xi = \frac{|S_{d-1}|}{|S_{d-2}|} \frac{1}{N(d, n)} \delta_{m,n}. \tag{11}$$

  Here $|S_{d-1}| = \frac{2\pi^{d/2}}{\Gamma(d/2)}$ denotes the surface of $S_{d-1}$, and $N(d, n)$ denotes the multiplicity of spherical harmonics of order $n$ on $S_{d-1}$, i.e. $N(d, n) = \frac{2n+d-2}{n} \binom{n+d-3}{n-1}$.

- This admits the orthogonal expansion of any analytic function $k(\xi)$ on $[-1, 1]$ into $P_n^d$ by

$$k(\xi) = \sum_{i=0}^{\infty} N(d, n) \frac{|S_{d-2}|}{|S_{d-1}|} P_n^d(\xi) \int_{-1}^{1} k(\xi') P_n^d(\xi')(1 - \xi'^2)^{\frac{d-3}{2}} d\xi'. \quad (12)$$

Moreover, the Legendre Polynomials may be expanded into an orthonormal basis of spherical harmonics $Y_{n,j}^d$ by the Funk-Hecke equation (cf. e.g. [4]) to obtain

$$P_n^d(x \cdot y) = \frac{|S_{d-1}|}{N(d, n)} \sum_{j=1}^{N(d,n)} Y_{n,j}^d(x) Y_{n,j}^d(y) \quad (13)$$

where $\|x\| = \|y\| = 1$ and moreover

$$\int_{S_{d-1}} Y_{n,j}^d(x) Y_{n',j'}^d(x) dx = \delta_{n,n'} \delta_{j,j'}. \quad (14)$$

## 4 Conditions and Eigensystems on $S_{d-1}$

Schoenberg [7] gives necessary and sufficient conditions under which a function $k(x \cdot y)$ defined on $S_{d-1}$ satisfies Mercer's condition. In particular he proves the following two theorems:

**Theorem 2 (Dot Product Kernels in Finite Dimensions)** *A kernel $k(x \cdot y)$ defined on $S_{d-1} \times S_{d-1}$ satisfies Mercer's condition if and only if its expansion into Legendre polynomials $P_n^d$ has only nonnegative coefficients, i.e.*

$$k(\xi) = \sum_{i=0}^{\infty} b_n P_n^d(\xi) \text{ with } b_n \geq 0. \quad (15)$$

**Theorem 3 (Dot Product Kernels in Infinite Dimensions)** *A kernel $k(x \cdot y)$ defined on the unit sphere in a Hilbert space satisfies Mercer's condition if and only if its Taylor series expansion has only nonnegative coefficients:*

$$k(\xi) = \sum_{i=0}^{\infty} a_n \xi^n \text{ with } a_n \geq 0. \quad (16)$$

Therefore, all we have to do in order to check whether a particular kernel may be used in a SV machine or a Gaussian Process is to look at its polynomial series expansion and check the coefficients. This will be done in Section 5.

Before doing so note that (16) is a more stringent condition than (15). In other words, in order to prove Mercer's condition for arbitrary dimensions it suffices to show that the Taylor expansion contains only positive coefficients. On the other hand, in order to prove that a candidate of a kernel function will never satisfy Mercer's condition, it is sufficient to show this for (15) where $P_n^d = P_n$, i.e. for the Legendre Polynomials.

We conclude this section with an explicit representation of the eigensystem of $k(x \cdot y)$. It is given by the following lemma:

**Lemma 4 (Eigensystem of Dot Product Kernels)** *Denote by $k(x \cdot y)$ a kernel on $S_{d-1} \times S_{d-1}$ satisfying condition (15) of Theorem 2. Then the eigensystem of $k$ is given by*

$$\Psi_{n,j} = Y_{n,j}^d \text{ with eigenvalues } \lambda_{n,j} = a_n \frac{|S_{d-1}|}{N(d,n)} \text{ of multiplicity } N(d,n). \quad (17)$$

*In other words, $\frac{a_n}{N(d,n)}$ determines the regularization properties of $k(x \cdot y)$.*

**Proof** Using the Funk-Hecke formula (13) we may expand (15) further into Spherical Harmonics $Y_{n,j}^d$. The latter, however, are orthonormal, hence computing the dot product of the resulting expansion with $Y_{n,j}^d(y)$ over $S_{d-1}$ leaves only the coefficient $Y_{n,j}^d(x) \frac{|S_{d-1}|}{N(d,n)}$ which proves that $Y_{n,j}^d$ are eigenfunctions of the integral operator $T_k$. ∎

In order to obtain the eigensystem of $k(x \cdot y)$ on $U_d$ we have to expand $k$ into $k(x \cdot y) = \sum_{m,n=0}^{\infty} (\|x\| \|y\|)^m P_n^d \left( \frac{x}{\|x\|} \cdot \frac{y}{\|y\|} \right)$ and expand $\Psi$ into $\Psi(\|x\|) \Psi \left( \frac{x}{\|x\|} \right)$. The latter is very technical and is thus omitted. See [6] for details.

## 5  Examples and Applications

In the following we will analyze a few kernels and state under which conditions they may be used as SV kernels.

**Example 1 (Homogeneous Polynomial Kernels $k(x,y) = (x \cdot y)^p$)** *It is well known that this kernel satisfies Mercer's condition for $p \in \mathbb{N}$. We will show that for $p \notin \mathbb{N}$ this is never the case.*

*Thus we have to show that (15) cannot hold for an expansion in terms of Legendre Polynomials $(d = 3)$. From [2, 7.126.1] we obtain for $k(x,y) = |\xi|^p$ (we need $|\xi|$ to make $k$ well-defined).*

$$\int_{-1}^{1} P_n(\xi) |\xi|^p d\xi = \frac{\sqrt{\pi} \Gamma(p+1)}{2^p \Gamma \left( 1 + \frac{p}{2} - \frac{n}{2} \right) \Gamma \left( \frac{3}{2} + \frac{p}{2} + \frac{n}{2} \right)} \text{ if } n \text{ even.} \quad (18)$$

*For odd $n$ the integral vanishes since $P_n(-\xi) = (-1)^n P_n(\xi)$. In order to satisfy (15), the integral has to be nonnegative for all $n$. One can see that $\Gamma \left( 1 + \frac{p}{2} - \frac{n}{2} \right)$ is the only term in (18) that may change its sign. Since the sign of the $\Gamma$ function alternates with period 1 for $x < 0$ (and has poles for negative integer arguments) we cannot find any $p$ for which $n = 2 \lfloor \frac{p}{2} + 1 \rfloor$ and $n = 2 \lceil \frac{p}{2} + 1 \rceil$ correspond to positive values of the integral.*

**Example 2 (Inhomogeneous Polynomial Kernels $k(x,y) = (x \cdot y + 1)^p$)**
*Likewise we might conjecture that $k(\xi) = (1 + \xi)^p$ is an admissible kernel for all $p > 0$. Again, we expand $k$ in a series of Legendre Polynomials to obtain [2, 7.127]*

$$\int_{-1}^{1} P_n(\xi)(\xi + 1)^p d\xi = \frac{2^{p+1} \Gamma^2(p+1)}{\Gamma(p+2+n) \Gamma(p+1-n)}. \quad (19)$$

*For $p \in \mathbb{N}$ all terms with $n > p$ vanish and the remainder is positive. For noninteger $p$, however, (19) may change its sign. This is due to $\Gamma(p+1-n)$. In particular, for any $p \notin \mathbb{N}$ (with $p > 0$) we have $\Gamma(p+1-n) < 0$ for $n = \lceil p \rceil + 1$. This violates condition (15), hence such kernels cannot be used in SV machines either.*

**Example 3 (Vovk's Real Polynomial $k(x,y) = \frac{1-(x \cdot y)^p}{1-(x \cdot y)}$ with $p \in \mathbb{N}$)** *This kernel can be written as $k(\xi) = \sum_{n=0}^{p-1} \xi^n$, hence all the coefficients $a_i = 1$ which means that this kernel can be used regardless of the dimensionality of the input space. Likewise we can analyze the an infinite power series:*

**Example 4 (Vovk's Infinite Polynomial $k(x,y) = (1-(x \cdot y))^{-1})$** *This kernel can be written as $k(\xi) = \sum_{n=0}^{\infty} \xi^n$, hence all the coefficients $a_i = 1$. It suggests poor generalization properties of that kernel.*

**Example 5 (Neural Networks Kernels $k(x,y) = \tanh(a + (x \cdot y)))$)** *It is a longstanding open question whether kernels $k(\xi) = \tanh(a + \xi)$ may be used as SV kernels, or, for which sets of parameters this might be possible. We show that is impossible for any set of parameters.*

*The technique is identical to the one of Examples 1 and 2: we have to show that $k$ fails the conditions of Theorem 2. Since this is very technical (and is best done by using computer algebra programs, e.g. Maple), we refer the reader to [6] for details and explain for the simpler case of Theorem 3 how the method works. Expanding $\tanh(a + \xi)$ into a Taylor series yields*

$$\tanh a + \xi \frac{1}{\cosh^2 a} - \xi^2 \frac{\tanh a}{\cosh^2 a} - \frac{\xi^3}{3}(1 - \tanh^2 a)(1 - 3\tanh^2 a) + O(\xi^4) \qquad (20)$$

*Now we analyze (20) coefficient-wise. Since all of them have to be nonnegative we obtain from the first term $a \in [0, \infty)$, the third term $a \in (-\infty, 0]$, and finally from the fourth term $|a| \in [\text{arctanh} \frac{1}{3}, \text{arctanh} \, 1]$. This leaves us with $a \in \emptyset$, hence under no conditions on its parameters the kernel above satisfies Mercer's condition.*

## 6   Eigensystems on $U_d$

In order to find the eigensystem of $T_k$ on $U_d$ we have to find a different representation of $k$ where the radial part $\|x\|\|y\|$ and the angular part $\xi = \left(\frac{x}{\|x\|} \cdot \frac{y}{\|y\|}\right)$ are factored out separately. We assume that $k(x \cdot y)$ can be written as

$$k(x \cdot y) = \sum_{n=0}^{\infty} \kappa_n(\|x\|\|y\|) P_n^d(\xi) \qquad (21)$$

where $\kappa_n$ are polynomials. To see that we can always find such an expansion for analytic functions, first expand $k$ in a Taylor series and then expand each coefficient $(\|x\|\|y\|\xi)^n$ into $(\|x\|\|y\|)^n \sum_{j=0}^{n} c_j(d,n) P_j^d(\xi)$. Rearranging terms into a series of $P_j^d$ gives expansion (21). This allows us to factorize the integral operator into its radial and its angular part. We obtain the following theorem:

**Theorem 5 (Eigenfunctions of $T_k$ on $U_d$)** *For any kernel $k$ with expansion (21) the eigensystem of the integral operator $T_k$ on $U_d$ is given by*

$$\Phi_{n,j,l}(x) = Y_{n,j}^d\left(\frac{x}{\|x\|}\right) \phi_{n,l}(\|x\|) \qquad (22)$$

*with eigenvalues $\Lambda_{n,j,l} = \frac{|S_{d-1}|}{N(d,n)} \lambda_{n,l}$, and multiplicity $N(d,n)$, where $(\phi_{n,l}, \lambda_{n,l})$ is the eigensystem of the integral operator*

$$\int_0^1 r_x^{d-1} \kappa_n(r_x r_y) \phi_{n,l}(r_x) dr_x = \lambda_{n,l} \phi_{n,l}(r_y). \qquad (23)$$

In general, (23) cannot be solved analytically. However, the accuracy of numerically solving (23) (finite integral in one dimension) is much higher than when diagonalizing $T_k$ directly.

**Proof** All we have to do is split the integral $\int_{U_d} dx$ into $\int_0^1 r^{d-1} dr \int_{S_{d-1}} d\Omega$. Moreover note that since $T_k$ commutes with the group of rotations it follows from group theory [4] that we may separate the angular and the radial part in the eigenfunctions, hence use the ansatz $\Phi(x) = \Phi_\Omega\left(\frac{x}{\|x\|}\right) \phi(\|x\|)$.

Next apply the Funk-Hecke equation (13) to expand the associated Legendre Polynomials $P_n^d$ into the spherical harmonics $Y_{n,j}^d$. As in Lemma 4 this leads to the spherical harmonics as the angular part of the eigensystem. The remaining radial part is then (23). See [6] for more details. ∎

This leads to the eigensystem of the homogeneous polynomial kernel $k(x,y) = (x \cdot y)^p$: if we use (18) in conjunction with (12) to expand $\xi^p$ into a series of $P_n^d(\xi)$ we obtain an expansion of type (21) where all $\kappa_n(r_x r_y) \propto (r_x r_y)^p$ for $n \le p$ and $\kappa_n(r_x r_y) = 0$ otherwise. Hence, the only solution to (23) is $\phi_n(r) = r^d$, thus $\Phi_{n,j}(x) = \|x\|^p Y_{n,j}^d\left(\frac{x}{\|x\|}\right)$. Eigenvalues can be obtained in a similar way.

# 7 Discussion

In this paper we gave conditions on the properties of dot product kernels, under which the latter satisfy Mercer's condition. While the requirements are relatively easy to check in the case where data is restricted to spheres (which allowed us to prove that several kernels never may be suitable SV kernels) and led to explicit formulations for eigenvalues and eigenfunctions, the corresponding calculations on balls are more intricate and mainly amenable to numerical analysis.

**Acknowledgments:** AS was supported by the DFG (Sm 62-1). The authors thank Bernhard Schölkopf for helpful discussions.

# References

[1] C. J. C. Burges. Geometry and invariance in kernel based methods. In B. Schölkopf, C. J. C. Burges, and A. J. Smola, editors, *Advances in Kernel Methods — Support Vector Learning*, pages 89–116, Cambridge, MA, 1999. MIT Press.

[2] I. S. Gradshteyn and I. M. Ryzhik. *Table of integrals, series, and products*. Academic Press, New York, 1981.

[3] J. Mercer. Functions of positive and negative type and their connection with the theory of integral equations. *Philos. Trans. Roy. Soc. London*, A 209:415–446, 1909.

[4] C. Müller. *Analysis of Spherical Symmetries in Euclidean Spaces*, volume 129 of *Applied Mathematical Sciences*. Springer, New York, 1997.

[5] N. Oliver, B. Schölkopf, and A.J. Smola. Natural regularization in SVMs. In A.J. Smola, P.L. Bartlett, B. Schölkopf, and D. Schuurmans, editors, *Advances in Large Margin Classifiers*, pages 51 – 60, Cambridge, MA, 2000. MIT Press.

[6] Z. Ovari. Kernels, eigenvalues and support vector machines. Honours thesis, Australian National University, Canberra, 2000.

[7] I. Schoenberg. Positive definite functions on spheres. *Duke Math. J.*, 9:96–108, 1942.

[8] A. Smola, B. Schölkopf, and K.-R. Müller. The connection between regularization operators and support vector kernels. *Neural Networks*, 11:637–649, 1998.

[9] G. Wahba. *Spline Models for Observational Data*, volume 59 of *CBMS-NSF Regional Conference Series in Applied Mathematics*. SIAM, Philadelphia, 1990.

[10] C. K. I. Williams. Prediction with Gaussian processes: From linear regression to linear prediction and beyond. In M. I. Jordan, editor, *Learning and Inference in Graphical Models*. Kluwer, 1998.
